# The Neurothermostat:
# Predictive Optimal Control of
# Residential Heating Systems

**Michael C. Mozer[†], Lucky Vidmar[†], Robert H. Dodier[‡]**
†Department of Computer Science
‡Department of Civil, Environmental, and Architectural Engineering
University of Colorado, Boulder, CO 80309-0430

## Abstract

The *Neurothermostat* is an adaptive controller that regulates indoor air temperature in a residence by switching a furnace on or off. The task is framed as an optimal control problem in which both comfort and energy costs are considered as part of the control objective. Because the consequences of control decisions are delayed in time, the Neurothermostat must anticipate heating demands with predictive models of occupancy patterns and the thermal response of the house and furnace. Occupancy pattern prediction is achieved by a hybrid neural net / look-up table. The Neurothermostat searches, at each discrete time step, for a decision sequence that minimizes the expected cost over a fixed planning horizon. The first decision in this sequence is taken, and this process repeats. Simulations of the Neurothermostat were conducted using artificial occupancy data in which regularity was systematically varied, as well as occupancy data from an actual residence. The Neurothermostat is compared against three conventional policies, and achieves reliably lower costs. This result is robust to the relative weighting of comfort and energy costs and the degree of variability in the occupancy patterns.

For over a quarter century, the home automation industry has promised to revolutionize our lifestyle with the so-called Smart House® in which appliances, lighting, stereo, video, and security systems are integrated under computer control. However, home automation has yet to make significant inroads, at least in part because software must be tailored to the home occupants.

Instead of expecting the occupants to program their homes or to hire someone to do so, one would ideally like the home to essentially *program itself* by observing the lifestyle of the occupants. This is the goal of the Neural Network House (Mozer et al., 1995), an actual residence that has been outfitted with over 75 sensors— including temperature, light, sound, motion—and actuators to control air heating, water heating, lighting, and ventilation. In this paper, we describe one research

project within the house, the *Neurothermostat*, that learns to regulate the indoor air temperature automatically by observing and detecting patterns in the occupants' schedules and comfort preferences. We focus on the problem of air heating with a whole-house furnace, but the same approach can be taken with alternative or multiple heating devices, and to the problems of cooling and ventilation.

# 1   TEMPERATURE REGULATION AS AN OPTIMAL CONTROL PROBLEM

Traditionally, the control objective of air temperature regulation has been to minimize energy consumption while maintaining temperature within an acceptable comfort margin during certain times of the day and days of the week. This is sensible in commercial settings, where occupancy patterns follow simple rules and where energy considerations dominate individual preferences. In a residence, however, the desires and schedules of occupants need to be weighted equally with energy considerations. Consequently, we frame the task of air temperature regulation as a problem of maximizing occupant comfort *and* minimizing energy costs.

These two objectives clearly conflict, but they can be integrated into a unified framework via an optimal control aproach in which the goal is to heat the house according to a policy that minimizes the cost

$$J = \lim_{\kappa \to \infty} \tfrac{1}{\kappa} \sum_{t=t_0+1}^{t_0+\kappa} \left[ e(u_t) + m(\mathbf{x}_t) \right],$$

where time, $t$, is quantized into nonoverlapping intervals during which we assume all environmental variables remain constant, $t_0$ is the interval ending at the current time, $u_t$ is the control decision for interval $t$ (e.g., turn the furnace on), $e(u)$ is the energy cost associated with decision $u$, $\mathbf{x}_t$ is the environmental state during interval $t$, which includes the indoor temperature and the occupancy status of the home, and $m(\mathbf{x})$ is the *misery* of the occupant given state $\mathbf{x}$. To add misery and energy costs, a common currency is required. Energy costs are readily expressed in dollars. We also determine misery in dollars, as we describe later.

While we have been unable to locate any earlier work that combined energy and comfort costs in an optimal control framework, optimal control has been used in a variety of building energy system control applications (e.g., Henze & Dodier, 1996; Khalid & Omatu, 1995).

# 2   THE NEUROTHERMOSTAT

Figure 1 shows the system architecture of the Neurothermostat and its interaction with the environment. The heart of the Neurothermostat is a controller that, at time intervals of $\delta$ minutes, can switch the house furnace on or off. Because the consequences of control decisions are delayed in time, the controller must be *predictive* to anticipate heating demands. The three boxes in the Figure depict components that predict or model various aspects of the environment. We explain their purpose via a formal description of the controller operation.

The controller considers sequences of $\kappa$ decisions, denoted $\mathbf{u}$, and searches for the sequence that minimizes the expected total cost, $\bar{J}_\mathbf{u}$, over the planning horizon of $\kappa\delta$ minutes:

$$\bar{J}_\mathbf{u} = \sum_{t=t_0+1}^{t_0+\kappa} E_{\mathbf{x}|\mathbf{u}} \left[ e(u_t) + m(\mathbf{x}_t) \right] = \sum_{t=t_0+1}^{t_0+\kappa} e(u_t) + \bar{m}_\mathbf{u}(\mathbf{x}_t),$$

where the expectation is computed over future states of the environment conditional on the decision sequence $\mathbf{u}$. The energy cost in an interval depends only on the control decision during that interval. The misery cost depends on two components

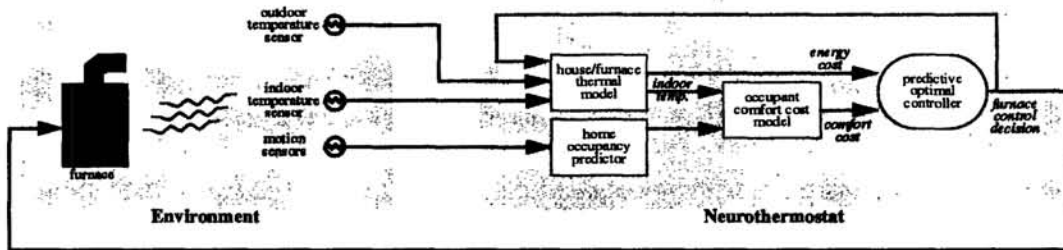

Figure 1: The Neurothermostat and its interaction with the environment

of the state—the house occupancy status, $o(t)$ (0 for empty, 1 for occupied), and the indoor air temperature, $h_{\mathbf{u}}(t)$:

$$\bar{m}_{\mathbf{u}}(o(t), h_{\mathbf{u}}(t)) = p[o(t) = 1]\, m(1, h_{\mathbf{u}}(t)) + p[o(t) = 0]\, m(0, h_{\mathbf{u}}(t))$$

Because the true quantities $e$, $h_{\mathbf{u}}$, $m$, and $p$ are unknown, they must be estimated. The house thermal model of Figure 1 provides $\hat{e}$ and $\hat{h}_{\mathbf{u}}$, the occupant comfort cost model provides $\hat{m}$, and the home occupancy predictor provides $\hat{p}$.

We follow a tradition of using neural networks for prediction in the context of building energy system control (e.g., Curtiss, Kreider, & Brandemuehl, 1993; Ferrano & Wong, 1990; Miller & Seem, 1991), although in our initial experiments we require a network only for the occupancy prediction.

## 2.1 PREDICTIVE OPTIMAL CONTROLLER

We propose a closed-loop controller that combines prediction with fixed-horizon planning, of the sort proposed by Clarke, Mohtadi, and Tuffs (1987). At each time step, the controller considers all possible decision sequences over the planning horizon and selects the sequence that minimizes the expected total cost, $\bar{J}$. The first decision in this sequence is then performed. After $\delta$ minutes, the planning and execution process is repeated. This approach assumes that beyond the planning horizon, all costs are independent of the first decision in the sequence.

While dynamic programming is an efficient search algorithm, it requires a discrete state space. Wishing to avoid quantizing the continuous variable of indoor temperature, and the errors that might be introduced, we performed performed exhaustive search through the possible decision sequences, which was tractable due to relatively short horizons and two additional domain constraints. First, because the house occupancy status can reasonably be assumed to be independent of the indoor temperature, $\hat{p}$ need not be recalculated for every possible decision sequence. Second, the current occupancy status depends on the recent occupancy history. Consequently, one needs to predict occupancy *patterns* over the planning horizon, $o \in \{0, 1\}^{\kappa}$ to compute $\hat{p}$. However, because most occupancy sequences are highly improbable, we find that considering only the most likely sequences—those containing at most two occupancy state transitions—produces the same decisions as doing the search over the entire distribution, reducing the cost from $O(2^{\kappa})$ to $O(\kappa^2)$.

## 2.2 OCCUPANCY PREDICTOR

The basic task of the occupancy predictor is to estimate the probability that the occupant will be home $\delta$ minutes in the future. The occupancy predictor can be run iteratively to estimate the probability of an occupancy pattern.

If occupants follow a deterministic daily schedule, a look up table indexed by time of day and current occupancy state should capture occupancy patterns. We thus use a look up table to encode whatever structure possible, and a neural network

to encode residual structure. The look up table divides time into fixed $\delta$ minute bins. The neural network consisted of the following inputs: current time of day; day of the week; average proportion of time home was occupied in the 10, 20, and 30 minutes from the present time of day on the previous three days and on the same day of the week during the past four weeks; and the proportion of time the home was occupied during the past 60, 180, and 360 minutes. The network, a standard three-layer architecture, was trained by back propagation. The number of hidden units was chosen by cross validation.

## 2.3  THERMAL MODEL OF HOUSE AND FURNACE

A thermal model of the house and furnace predicts future indoor temperature(s) as a function of the outdoor temperature and the furnace operation. While one could perform system identification using neural networks, a simple parameterized resistance-capacitance (RC) model provides a reasonable first-order approximation. The RC model assumes that: the inside of the house is at a uniform temperature, and likewise the outside; a homogeneous flat wall separates the inside and outside, and this wall has a thermal resistance $R$ and thermal capacitance $C$; the entire wall mass is at the inside temperature; and the heat input to the inside is $Q$ when the furnace is running or zero otherwise. Assuming that the outdoor temperature, denoted $g$, is constant, the the indoor temperature at time $t$, $\hat{h}_{\mathbf{u}}(t)$, is:

$$\hat{h}_{\mathbf{u}}(t) = \hat{h}_{\mathbf{u}}(t-1)\exp(-60\delta/RC) + (RQu(t) + g)(1 - \exp(-60\delta/RC)),$$

where $\hat{h}_{\mathbf{u}}(t_0)$ is the actual indoor temperature at the current time. $R$ and $C$ were determined by architectural properties of the Neural Network House to be 1.33 Kelvins/kilowatt and 16 megajoules/Kelvin, respectively. The House furnace is rated at 133,000 Btu/hour and has 92.5% fuel use efficiency, resulting in an output of $Q = 36.1$ kilowatts. With natural gas at \$.485 per CCF, the cost of operating the furnace, $\hat{e}$, is \$.7135 per hour.

## 2.4  OCCUPANT COMFORT COST MODEL

In the Neural Network House, the occupant expresses discomfort by adjusting a setpoint temperature on a control panel. However, for simplicity, we assume in this work the setpoint is a constant, $\lambda$. When the home is occupied, the misery cost is a monotonic function of the deviation of the actual indoor temperature from the setpoint. When the home is empty, the misery cost is zero regardless of the temperature.

There is a rich literature directed at measuring thermal comfort in a given environment (i.e., dry-bulb temperature, relative humidity, air velocity, clothing insulation, etc.) for the average building occupant (e.g., Fanger, 1972; Gagge, Stolwijk, & Nishi, 1971). Although the measurements indicate the fraction of the population which is uncomfortable in a particular environment, one might also interpret them as a measure of an individual's level of discomfort. As a function of dry-bulb temperature, this curve is roughly parabolic. We approximate it with a measure of misery in a $\delta$-minute period as follows:

$$\hat{m}(o, h) = o\alpha \frac{\delta}{24 \times 60} \frac{\max(0, |\lambda - h| - \epsilon)^2}{25}.$$

The first term, $o$, is a binary variable indicating the home occupancy state. The second term is a conversion factor from arbitrary "misery" units to dollars. The third term scales the misery cost from a full day to the basic update interval. The fourth term produces the parabolic relative misery function, scaled so that a temperature difference of $5°$ C produces one unit of misery, with a *deadband* region from $\lambda - \epsilon$ to $\lambda + \epsilon$.

We have chosen the conversion factor $\alpha$ using an economic perspective. Consider the lost productivity that results from trying to work in a home that is $5°$ C colder

than desired for a 24 hour period. Denote this loss $\rho$, measured in hours. The cost in dollars of this loss is then $\alpha = \gamma\rho$, where $\gamma$ is the individual's hourly salary. With this approch, $\alpha$ can be set in a natural, intuitive manner.

## 3   SIMULATION METHODOLOGY

In all experiments we report below, $\delta = 10$ minutes, $\kappa = 12$ steps (120 minute planning horizon), $\lambda = 22.5°$ C, $\epsilon = 1$, and $\gamma = 28$ dollars per hour. The productivity loss, $\rho$, was varied from 1 to 3 hours.

We report here results from the Neurothermostat operating in a simulated environment, rather than in the actual Neural Network House. The simulated environment incorporates the house/furnace thermal model described earlier and occupants whose preferences follow the comfort cost model. The outdoor temperature is assumed to remain a constant $0°$ C. Thus, the Neurothermostat has an accurate model of its environment, except for the occupancy patterns, which it must predict based on training data. This allows us to evaluate the performance of the Neurothermostat and the occupancy model as occupancy patterns are varied, uncontaminated by the effect of inaccuracy in the other internal models.

We have evaluated the Neurothermostat with both real and artificial occupancy data. The real data was collected from the Neural Network House with a single resident over an eight month period, using a simple algorithm based on motion detector output and the opening and closing of outside doors. The artificial data was generated by a simulation of a single occupant. The occupant would go to work each day, later on the weekends, would sometimes come home for lunch, sometimes go out on weekend nights, and sometimes go out of town for several days. To examine performance of the Neurothermostat as a function of the variability in the occupant's schedule, the simulation model included a parameter, the *variability index*. An index of 0 means that the schedule is entirely deterministic; an index of 1 means that the schedule was very noisy, but still contained statistical regularities. The index determined factors such as the likelihood and duration of out-of-town trips and the variability in departure and return times.

### 3.1   ALTERNATIVE HEATING POLICIES

In addition to the Neurothermostat, we examined three nonadaptive control policies. These policies produce a *setpoint* at each time step, and the furnace is switched on if the temperature drops below the setpoint and off if the temperature rises above the setpoint. (We need not be concerned about damage to the furnace by cycling because control decisions are made ten minutes apart.) The *constant-temperature* policy produces a fixed setpoint of $22.5°$ C. The *occupancy-triggered* policy produces a setpoint of $18°$ C when the house is empty, $22.5°$ C when the house is occupied. The *setback-thermostat* policy lowers the setpoint from $22.5°$ C to $18°$ C half an hour before the mean work departure time for that day of the week, and raises it back to $22.5°$ C half an hour before the mean work return time for that day of the week. The setback temperature for the occupancy-triggered and setback-thermostat policies was determined empirically to minimize the total cost.

## 4   RESULTS

### 4.1   OCCUPANCY PREDICTOR

Performance of three different predictors was evaluated using artificial data across a range of values for the variability index. For each condition, we generated eight training/test sets of artificial data, each training and test set consisting of 150 days of data. Table 1 shows the normalized mean squared error (MSE) on the test set, averaged over the eight replications. The normalization involved dividing the MSE for each replication by the error obtained by predicting the future occupancy state

Table 1: Normalized MSE on Test Set for Occupancy Prediction—Artificial Data

| | variability index | | | | |
|---|---|---|---|---|---|
| | 0 | .25 | .50 | .75 | 1 |
| *lookup table* | .49 | .81 | .94 | .92 | .94 |
| *neural net* | .02 | .63 | .83 | .86 | .91 |
| *lookup table + neural net* | .02 | .60 | .78 | .77 | .74 |

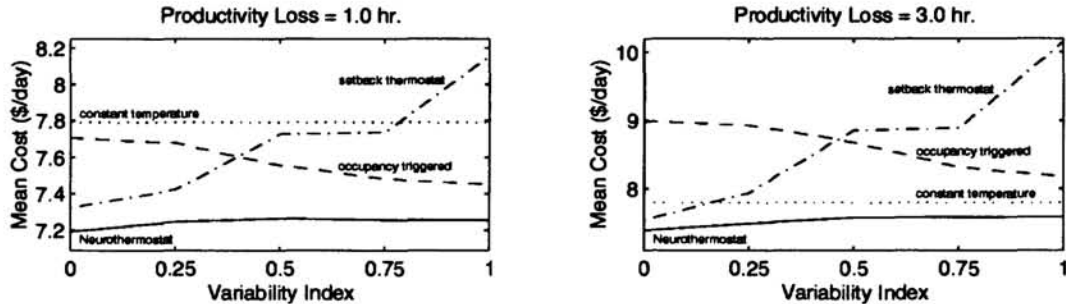

Figure 2: Mean cost per day incurred by four control policies on (artificial) test data as a function of the data's variability index for $\rho = 1$ (comfort lightly weighted, left panel) and $\rho = 3$ (comfort heavily weighted, right panel).

is the same as the present state. The main result here is that the combination of neural network and look up table perform better than either component in isolation (ANOVA: $F(1,7) = 1121, p < .001$ for combination vs. table; $F(1,7) = 64, p < .001$ for combination vs. network), indicating that the two components are capturing different structure in the data.

### 4.2  CONTROLLER WITH ARTIFICIAL OCCUPANCY DATA

Having trained eight occupancy predictors with different (artificial data) training sets, we computed misery and energy costs for the Neurothermostat on the corresponding test sets. Figure 2 shows the mean total cost per day as a function of variability index, control policy, and relative comfort cost. The robust result is that the Neurothermostat outperforms the three nonadaptive control policies for all levels of the variability index and for both a wide range of values of $\rho$.

Other patterns in the data are noteworthy. Costs for the Neurothermostat tend to rise with the variability index, as one would expect because the occupant's schedule becomes less predictable. The constant-temperature policy is worst if occupant comfort is weighted lightly, and begins to approach the Neurothermostat in performance as comfort costs are increased. If comfort costs overwhelm energy costs, then the constant-temperature policy and the Neurothermostat converge.

### 4.3  CONTROLLER WITH REAL OCCUPANCY DATA

Eight months of real occupancy data collected in the Neural Network House beginning in September 1994 was also used to generate occupancy models and test controllers. Three training/test splits were formed by training on five consecutive months and testing on the next month. Table 2 shows the mean daily cost for the four controllers. The Neurothermostat significantly outperforms the three nonadaptive controllers, as it did with the artificial data.

### 5  DISCUSSION

The simulation studies reported here strongly suggest that adaptive control of residential heating and cooling systems is worthy of further investigation. One is

Table 2: Mean Daily Cost Based on Real Occupancy Data

|                      | productivity loss | |
|                      | $\rho = 1$ | $\rho = 3$ |
| --- | --- | --- |
| *Neurothermostat*    | $6.77 | $7.05 |
| *constant temperature* | $7.85 | $7.85 |
| *occupancy triggered* | $7.49 | $8.66 |
| *setback thermostat* | $8.12 | $9.74 |

tempted to trumpet the conclusion that adaptive control lowers heating costs, but before doing so, one must be clear that the cost being lowered is a combination of comfort and energy costs. If one is merely interested in lowering energy costs, then simply shut off the furnace. A central contribution of this work is thus the framing of the task of air temperature regulation as an optimal control problem in which both comfort and energy costs are considered as part of the control objective.

A common reaction to this research project is, "My life is far too irregular to be predicted. I don't return home from work at the same time every day." An important finding of this work is that even a highly nondeterministic schedule contains sufficient statistical regularity to be exploited by a predictive controller. We found this for both artificial data with a high variability index and real occupancy data.

A final contribution of our work is to show that for periodic data such as occupancy patterns that follow a weekly schedule, the combination of a look up table to encode the periodic structure and a neural network to encode the residual structure can outperform either method in isolation.

## Acknowledgements

Support for this research has come from Lifestyle Technologies, NSF award IRI-9058450, and a CRCW grant-in-aid from the University of Colorado. This project owes its existence to the dedication of many students, particularly Marc Anderson, Josh Anderson, Paul Kooros, and Charles Myers. Our thanks to Reid Hastie and Gary McClelland for their suggestions on assessing occupant misery.

## References

Clarke, D. W., Mohtadi, C., & Tuffs, P. S. (1987). Generalized predictive control–Part I. The basic algorithm. *Automatica*, *23*, 137–148.

Curtiss, P., Kreider, J. F., & Brandemuehl, M. J. (1993). Local and global control of commercial building HVAC systems using artificial neural networks. *Proceedings of the American Control Conference, 3*, 3029–3044.

Fanger, P. O. (1972). *Thermal comfort.* New York: McGraw-Hill.

Ferrano, F. J., & Wong, K. V. (1990). Prediction of thermal storage loads using a neural network. *ASHRAE Transactions, 96*, 723–726.

Gagge, A. P., Stolwijk, J. A. J., & Nishi, Y. (1971). An effective temperature scale based on a simple model of human physiological regulatory response. *ASHRAE Transactions, 77*, 247–262.

Henze, G. P., & Dodier, R. H. (1996). Development of a predictive optimal controller for thermal energy storage systems. Submitted for publication.

Khalid, M., & Omatu, S. (1995). Temperature regulation with neural networks and alternative control schemes. *IEEE Transactions on Neural Networks, 6*, 572–582.

Miller, R. C., & Seem, J. E. (1991). Comparison of artificial neural networks with traditional methods of predicting return time from night or weekend setback. *ASHRAE Transactions, 97*, 500–508.

Mozer, M. C., Dodier, R. H., Anderson, M., Vidmar, L., Cruickshank III, R. F., & Miller, D. (1995). The neural network house: An overview. In L. Niklasson & M. Boden (Eds.), *Current trends in connectionism* (pp. 371–380). Hillsdale, NJ: Erlbaum.